# On the Convergence of the Concave-Convex Procedure

**Bharath K. Sriperumbudur**
Department of Electrical and Computer Engineering
University of California, San Diego
La Jolla, CA 92093
bharathsv@ucsd.edu

**Gert R. G. Lanckriet**
Department of Electrical and Computer Engineering
University of California, San Diego
La Jolla, CA 92093
gert@ece.ucsd.edu

## Abstract

The concave-convex procedure (CCCP) is a majorization-minimization algorithm that solves d.c. (difference of convex functions) programs as a sequence of convex programs. In machine learning, CCCP is extensively used in many learning algorithms like sparse support vector machines (SVMs), transductive SVMs, sparse principal component analysis, etc. Though widely used in many applications, the convergence behavior of CCCP has not gotten a lot of specific attention. Yuille and Rangarajan analyzed its convergence in their original paper, however, we believe the analysis is not complete. Although the convergence of CCCP can be derived from the convergence of the d.c. algorithm (DCA), its proof is more specialized and technical than actually required for the specific case of CCCP. In this paper, we follow a different reasoning and show how Zangwill's *global convergence* theory of iterative algorithms provides a natural framework to prove the convergence of CCCP, allowing a more elegant and simple proof. This underlines Zangwill's theory as a powerful and general framework to deal with the convergence issues of iterative algorithms, after also being used to prove the convergence of algorithms like expectation-maximization, generalized alternating minimization, etc. In this paper, we provide a rigorous analysis of the convergence of CCCP by addressing these questions: (i) When does CCCP find a local minimum or a stationary point of the d.c. program under consideration? (ii) When does the sequence generated by CCCP converge? We also present an open problem on the issue of *local convergence* of CCCP.

## 1 Introduction

The concave-convex procedure (CCCP) [30] is a majorization-minimization algorithm [15] that is popularly used to solve d.c. (difference of convex functions) programs of the form,

$$\min_{x} \quad f(x)$$
$$\text{s.t.} \quad c_i(x) \leq 0, \ i \in [m],$$
$$d_j(x) = 0, \ j \in [p], \tag{1}$$

where $f(x) = u(x) - v(x)$ with $u$, $v$ and $c_i$ being real-valued convex functions, $d_j$ being an affine function, all defined on $\mathbb{R}^n$. Here, $[m] := \{1, \ldots, m\}$. Suppose $v$ is differentiable. The CCCP

algorithm is an iterative procedure that solves the following sequence of convex programs,

$$x^{(l+1)} \in \arg\min_x \quad u(x) - x^T \nabla v(x^{(l)})$$

$$\text{s.t.} \quad c_i(x) \le 0, \ i \in [m],$$
$$d_j(x) = 0, \ j \in [p]. \tag{2}$$

As can be seen from (2), the idea of CCCP is to linearize the concave part of $f$, which is $-v$, around a solution obtained in the current iterate so that $u(x) - x^T \nabla v(x^{(l)})$ is convex in $x$, and therefore the non-convex program in (1) is solved as a sequence of convex programs as shown in (2). The original formulation of CCCP by Yuille and Rangarajan [30] deals with unconstrained and linearly constrained problems. However, the same formulation can be extended to handle any constraints (both convex and non-convex). CCCP has been extensively used in solving many non-convex programs (of the form in (1)) that appear in machine learning. For example, [3] proposed a successive linear approximation (SLA) algorithm for feature selection in support vector machines, which can be seen as a special case of CCCP. Other applications where CCCP has been used include sparse principal component analysis [27], transductive SVMs [11, 5, 28], feature selection in SVMs [22], structured estimation [10], missing data problems in Gaussian processes and SVMs [26], etc.

The algorithm in (2) starts at some random point $x^{(0)} \in \{x : c_i(x) \le 0, \ i \in [m]; \ d_j(x) = 0, \ j \in [p]\}$, solves the program in (2) and therefore generates a sequence $\{x^{(l)}\}_{l=0}^{\infty}$. The goal of this paper is to study the convergence of $\{x^{(l)}\}_{l=0}^{\infty}$: (i) When does CCCP find a local minimum or a stationary point[1] of the program in (1)? (ii) Does $\{x^{(l)}\}_{l=0}^{\infty}$ converge? If so, to what and under what conditions? From a practical perspective, these questions are highly relevant, given that CCCP is widely applied in machine learning.

In their original CCCP paper, Yuille and Rangarajan [30, Theorem 2] analyzed its convergence, but we believe the analysis is not complete. They showed that $\{x^{(l)}\}_{l=0}^{\infty}$ satisfies the monotonic descent property, i.e., $f(x^{(l+1)}) \le f(x^{(l)})$ and argued that this descent property ensures the convergence of $\{x^{(l)}\}_{l=0}^{\infty}$ to a minimum or saddle point of the program in (1). However, a rigorous proof is not provided, to ensure that their claim holds for all $u$, $v$, $\{c_i\}$ and $\{d_j\}$. Answering the previous questions, however, requires a rigorous proof of the convergence of CCCP that explicitly mentions the conditions under which it can happen.

In the d.c. programming literature, Pham Dinh and Hoai An [8] proposed a primal-dual subdifferential method called DCA (d.c. algorithm) for solving a general d.c. program of the form $\min\{u(x) - v(x) : x \in \mathbb{R}^n\}$, where it is assumed that $u$ and $v$ are proper lower semi-continuous convex functions, which form a larger class of functions than the class of differentiable functions. It can be shown that if $v$ is differentiable, then DCA exactly reduces to CCCP. Unlike in CCCP, DCA involves constructing two sets of convex programs (called the primal and dual programs) and solving them iteratively in succession such that the solution of the primal is the initialization to the dual and vice-versa. See [8] for details. [8, Theorem 3] proves the convergence of DCA for general d.c. programs. The proof is specialized and technical. It fundamentally relies on d.c. duality, however, outlining the proof in any more detail requires a substantial discussion which would lead us too far here. In this work, we follow a fundamentally different approach and show that the convergence of CCCP, specifically, can be analyzed in a more simple and elegant way, by relying on Zangwill's *global convergence* theory of iterative algorithms. We make some simple assumptions on the functions involved in (1), which are not too restrictive and therefore applicable to many practical situations. The tools employed in our proof are of completely different flavor than the ones used in the proof of DCA convergence: DCA convergence analysis exploits d.c. duality while we use the notion of point-to-set maps as introduced by Zangwill. Zangwill's theory is a powerful and general framework to deal with the convergence issues of iterative algorithms. It has also been used to prove the convergence of the expectation-maximization (EM) algorithm [29], generalized alternating minimization algorithms [12], multiplicative updates in non-negative quadratic programming [25], etc. and is therefore a natural framework to analyze the convergence of CCCP in a more direct way.

The paper is organized as follows. In Section 2, we provide a brief introduction to majorization-minimization (MM) algorithms and show that CCCP is obtained as a particular form of majorization-

minimization. The goal of this section is also to establish the literature on MM algorithms and show where CCCP fits in it. In Section 3, we present Zangwill's theory of global convergence, which is a general framework to analyze the convergence behavior of iterative algorithms. This theory is used to address the *global convergence* of CCCP in Section 4. This involves analyzing the *fixed points* of the CCCP algorithm in (2) and then showing that the fixed points are the stationary points of the program in (1). The results in Section 4 are extended in Section 4.1 to analyze the convergence of the *constrained concave-convex procedure* that was proposed by [26] to deal with d.c. programs with d.c. constraints. We briefly discuss the *local convergence* issues of CCCP in Section 5 and conclude the section with an open question.

## 2   Majorization-minimization

MM algorithms can be thought of as a generalization of the well-known EM algorithm [7]. The general principle behind MM algorithms was first enunciated by the numerical analysts, Ortega and Rheinboldt [23] in the context of line search methods. The MM principle appears in many places in statistical computation, including multidimensional scaling [6], robust regression [14], correspondence analysis [13], variable selection [16], sparse signal recovery [4], etc. We refer the interested reader to a tutorial on MM algorithms [15] and the references therein.

The general idea of MM algorithms is as follows. Suppose we want to minimize $f$ over $\Omega \subset \mathbb{R}^n$. The idea is to construct a *majorization function* $g$ over $\Omega \times \Omega$ such that

$$\begin{cases} f(x) \le g(x,y), \ \forall \, x,y \in \Omega \\ f(x) = g(x,x), \ \forall \, x \in \Omega \end{cases}. \tag{3}$$

Thus, $g$ as a function of $x$ is an upper bound on $f$ and coincides with $f$ at $y$. The majorization algorithm corresponding with this majorization function $g$ updates $x$ at iteration $l$ by

$$x^{(l+1)} \in \arg\min_{x \in \Omega} g(x, x^{(l)}), \tag{4}$$

unless we already have $x^{(l)} \in \arg\min_{x \in \Omega} g(x, x^{(l)})$, in which case the algorithm stops. The majorization function, $g$ is usually constructed by using Jensen's inequality for convex functions, the first-order Taylor approximation or the quadratic upper bound principle [1]. However, any other method can also be used to construct $g$ as long as it satisfies (3). It is easy to show that the above iterative scheme decreases the value of $f$ monotonically in each iteration, i.e.,

$$f(x^{(l+1)}) \le g(x^{(l+1)}, x^{(l)}) \le g(x^{(l)}, x^{(l)}) = f(x^{(l)}), \tag{5}$$

where the first inequality and the last equality follow from (3) while the sandwiched inequality follows from (4).

Note that MM algorithms can be applied equally well to the maximization of $f$ by simply reversing the inequality sign in (3) and changing the "min" to "max" in (4). In this case, the word MM refers to minorization-maximization, where the function $g$ is called the *minorization function*. To put things in perspective, the EM algorithm can be obtained by constructing the minorization function $g$ using Jensen's inequality for concave functions. The construction of such a $g$ is referred to as the E-step, while (4) with the "min" replaced by "max" is referred to as the M-step. The algorithm in (3) and (4) is also referred to as the *auxiliary function method*, e.g., for non-negative matrix factorization [18]. [17] studied this algorithm under the name *optimization transfer* while [19] referred to it as the SM algorithm, where "S" stands for the surrogate step (same as the majorization/minorization step) and "M" stands for the minimization/maximization step depending on the problem at hand. $g$ is called the surrogate function. In the following example, we show that CCCP is an MM algorithm for a particular choice of the majorization function, $g$.

**Example 1** (Linear Majorization)**.** *Let us consider the optimization problem,* $\min_{x \in \Omega} f(x)$ *where* $f = u - v$, *with $u$ and $v$ both real-valued, convex, defined on $\mathbb{R}^n$ and $v$ differentiable. Since $v$ is convex, we have* $v(x) \ge v(y) + (x-y)^T \nabla v(y), \ \forall \, x,y \in \Omega$. *Therefore,*

$$f(x) \le u(x) - v(y) - (x-y)^T \nabla v(y) =: g(x,y). \tag{6}$$

*It is easy to verify that $g$ is a majorization function of $f$. Therefore, we have*

$$\begin{aligned} x^{(l+1)} \ &\in \ \arg\min_{x \in \Omega} g(x, x^{(l)}) \\ &= \ \arg\min_{x \in \Omega} u(x) - x^T \nabla v(x^{(l)}). \end{aligned} \tag{7}$$

*If $\Omega$ is a convex set, then the above procedure reduces to CCCP, which solves a sequence of convex programs. As mentioned before, CCCP is proposed for unconstrained and linearly constrained non-convex programs. This example shows that the same idea can be extended to any constraint set.*

*Suppose $u$ and $v$ are strictly convex, then a strict descent can be achieved in (5) unless $x^{(l+1)} = x^{(l)}$, i.e., if $x^{(l+1)} \neq x^{(l)}$, then*

$$f(x^{(l+1)}) < g(x^{(l+1)}, x^{(l)}) < g(x^{(l)}, x^{(l)}) = f(x^{(l)}). \tag{8}$$

*The first strict inequality follows from (6). The strict convexity of $u$ leads to the strict convexity of $g$ and therefore $g(x^{(l+1)}, x^{(l)}) < g(x^{(l)}, x^{(l)})$ unless $x^{(l+1)} = x^{(l)}$.*

# 3 Global convergence theory of iterative algorithms

For an iterative procedure like CCCP to be useful, it must converge to a local optimum or a stationary point from all or at least a significant number of initialization states and not exhibit other nonlinear system behaviors, such as divergence or oscillation. This behavior can be analyzed by using the global convergence theory of iterative algorithms developed by Zangwill [31]. Note that the word "global convergence" is a misnomer. We will clarify it below and also introduce some notation and terminology.

To understand the convergence of an iterative procedure like CCCP, we need to understand the notion of a *set-valued mapping*, or *point-to-set mapping*, which is central to the theory of global convergence.[2] A point-to-set map $\Psi$ from a set $X$ into a set $Y$ is defined as $\Psi : X \rightarrow \mathscr{P}(Y)$, which assigns a subset of $Y$ to each point of $X$, where $\mathscr{P}(Y)$ denotes the power set of $Y$. We introduce few definitions related to the properties of point-to-set maps that will be used later. Suppose $X$ and $Y$ are two topological spaces. A point-to-set map $\Psi$ is said to be *closed* at $x_0 \in X$ if $x_k \rightarrow x_0$ as $k \rightarrow \infty$, $x_k \in X$ and $y_k \rightarrow y_0$ as $k \rightarrow \infty$, $y_k \in \Psi(x_k)$, imply $y_0 \in \Psi(x_0)$. This concept of *closure* generalizes the concept of continuity for ordinary point-to-point mappings. A point-to-set map $\Psi$ is said to be closed on $S \subset X$ if it is closed at every point of $S$. A *fixed point* of the map $\Psi : X \rightarrow \mathscr{P}(X)$ is a point $x$ for which $\{x\} = \Psi(x)$, whereas a *generalized fixed point* of $\Psi$ is a point for which $x \in \Psi(x)$. $\Psi$ is said to be *uniformly compact* on $X$ if there exists a compact set $H$ independent of $x$ such that $\Psi(x) \subset H$ for all $x \in X$. Note that if $X$ is compact, then $\Psi$ is uniformly compact on $X$. Let $\phi : X \rightarrow \mathbb{R}$ be a continuous function. $\Psi$ is said to be *monotonic* with respect to $\phi$ whenever $y \in \Psi(x)$ implies that $\phi(y) \leq \phi(x)$. If, in addition, $y \in \Psi(x)$ and $\phi(y) = \phi(x)$ imply that $y = x$, then we say that $\Psi$ is *strictly monotonic*.

Many iterative algorithms in mathematical programming can be described using the notion of point-to-set maps. Let $X$ be a set and $x_0 \in X$ a given point. Then an *algorithm*, $\mathcal{A}$, with initial point $x_0$ is a point-to-set map $\mathcal{A} : X \rightarrow \mathscr{P}(X)$ which generates a sequence $\{x_k\}_{k=1}^{\infty}$ via the rule $x_{k+1} \in \mathcal{A}(x_k)$, $k = 0, 1, \ldots$. $\mathcal{A}$ is said to be *globally convergent* if *for any chosen initial point $x_0$*, the sequence $\{x_k\}_{k=0}^{\infty}$ generated by $x_{k+1} \in \mathcal{A}(x_k)$ (or a subsequence) converges to a point for which a necessary condition of optimality holds. The property of global convergence expresses, in a sense, the certainty that the algorithm works. It is very important to stress the fact that it does not imply (contrary to what the term might suggest) convergence to a global optimum for all initial points $x_0$.

With the above mentioned concepts, we now state Zangwill's global convergence theorem [31, Convergence theorem A, page 91].

**Theorem 2** ([31]). *Let $\mathcal{A} : X \rightarrow \mathscr{P}(X)$ be a point-to-set map (an algorithm) that given a point $x_0 \in X$ generates a sequence $\{x_k\}_{k=0}^{\infty}$ through the iteration $x_{k+1} \in \mathcal{A}(x_k)$. Also let a solution set $\Gamma \subset X$ be given. Suppose*

*(1) All points $x_k$ are in a compact set $S \subset X$.*

*(2) There is a continuous function $\phi : X \rightarrow \mathbb{R}$ such that:*

*(a) $x \notin \Gamma \Rightarrow \phi(y) < \phi(x), \forall y \in \mathcal{A}(x)$,*

*(b)* $x \in \Gamma \Rightarrow \phi(y) \leq \phi(x), \forall\, y \in \mathcal{A}(x)$.

*(3)* $\mathcal{A}$ *is closed at* $x$ *if* $x \notin \Gamma$.

*Then the limit of any convergent subsequence of* $\{x_k\}_{k=0}^{\infty}$ *is in* $\Gamma$*. Furthermore,* $\lim_{k \to \infty} \phi(x_k) = \phi(x_*)$ *for all limit points* $x_*$.

The general idea in showing the global convergence of an algorithm, $\mathcal{A}$ is to invoke Theorem 2 by appropriately defining $\phi$ and $\Gamma$. For an algorithm $\mathcal{A}$ that solves the minimization problem, $\min\{f(x) : x \in \Omega\}$, the solution set, $\Gamma$ is usually chosen to be the set of corresponding stationary points and $\phi$ can be chosen to be the objective function itself, i.e., $f$, if $f$ is continuous. In Theorem 2, the convergence of $\phi(x_k)$ to $\phi(x_*)$ does not automatically imply the convergence of $x_k$ to $x_*$. However, if $\mathcal{A}$ is strictly monotone with respect to $\phi$, then Theorem 2 can be strengthened by using the following result due to Meyer [20, Theorem 3.1, Corollary 3.2].

**Theorem 3** ([20]). *Let* $\mathcal{A} : X \to \mathscr{P}(X)$ *be a point-to-set map such that* $\mathcal{A}$ *is uniformly compact, closed and strictly monotone on* $X$*, where* $X$ *is a closed subset of* $\mathbb{R}^n$*. If* $\{x_k\}_{k=0}^{\infty}$ *is any sequence generated by* $\mathcal{A}$*, then all limit points will be fixed points of* $\mathcal{A}$*,* $\phi(x_k) \to \phi(x_*) =: \phi^*$ *as* $k \to \infty$*, where* $x_*$ *is a fixed point,* $\|x_{k+1} - x_k\| \to 0$*, and either* $\{x_k\}_{k=0}^{\infty}$ *converges or the set of limit points of* $\{x_k\}_{k=0}^{\infty}$ *is connected. Define* $\mathscr{F}(a) := \{x \in \mathscr{F} : \phi(x) = a\}$ *where* $\mathscr{F}$ *is the set of fixed points of* $\mathcal{A}$*. If* $\mathscr{F}(\phi^*)$ *is finite, then any sequence* $\{x_k\}_{k=0}^{\infty}$ *generated by* $\mathcal{A}$ *converges to some* $x_*$ *in* $\mathscr{F}(\phi^*)$.

Both these results just use basic facts of analysis and are simple to prove and understand. Using these results on the global convergence of algorithms, [29] has studied the convergence properties of the EM algorithm, while [12] analyzed the convergence of generalized alternating minimization procedures. In the following section, we use these results to analyze the convergence of CCCP.

## 4 Convergence theorems for CCCP

Let us consider the CCCP algorithm in (2) pertaining to the d.c. program in (1). Let $\mathcal{A}_{cccp}$ be the point-to-set map, $x^{(l+1)} \in \mathcal{A}_{cccp}(x^{(l)})$ such that

$$\mathcal{A}_{cccp}(y) = \arg\min\{u(x) - x^T \nabla v(y) \,:\, x \in \Omega\}, \tag{9}$$

where $\Omega := \{x \,:\, c_i(x) \leq 0,\, i \in [m],\, d_j(x) = 0,\, j \in [p]\}$. Let us assume that $\{c_i\}$ are differentiable convex functions defined on $\mathbb{R}^n$. We now present the global convergence theorem for CCCP.

**Theorem 4** (Global convergence of CCCP$-$I). *Let* $u$ *and* $v$ *be real-valued differentiable convex functions defined on* $\mathbb{R}^n$*. Suppose* $\nabla v$ *is continuous. Let* $\{x^{(l)}\}_{l=0}^{\infty}$ *be any sequence generated by* $\mathcal{A}_{cccp}$ *defined by (9). Suppose* $\mathcal{A}_{cccp}$ *is uniformly compact[3] on* $\Omega$ *and* $\mathcal{A}_{cccp}(x)$ *is nonempty for any* $x \in \Omega$*. Then, assuming suitable constraint qualification, all the limit points of* $\{x^{(l)}\}_{l=0}^{\infty}$ *are stationary points of the d.c. program in (1). In addition* $\lim_{l \to \infty}(u(x^{(l)}) - v(x^{(l)})) = u(x_*) - v(x_*)$*, where* $x_*$ *is some stationary point of* $\mathcal{A}_{cccp}$.

Before we proceed with the proof of Theorem 4, we need a few additional results. The idea of the proof is to show that any generalized fixed point of $\mathcal{A}_{cccp}$ is a stationary point of (1), which is shown below in Lemma 5, and then use Theorem 2 to analyze the generalized fixed points.

**Lemma 5.** *Suppose* $x_*$ *is a generalized fixed point of* $\mathcal{A}_{cccp}$ *and assume that constraints in (9) are qualified at* $x_*$*. Then,* $x_*$ *is a stationary point of the program in (1).*

*Proof.* We have $x_* \in \mathcal{A}_{cccp}(x_*)$ and the constraints in (9) are qualified at $x_*$. Then, there exists Lagrange multipliers $\{\eta_i^*\}_{i=1}^{m} \subset \mathbb{R}_+$ and $\{\mu_j^*\}_{j=1}^{p} \subset \mathbb{R}$ such that the following KKT conditions hold:

$$\begin{cases} \nabla u(x_*) - \nabla v(x_*) + \sum_{i=1}^{m} \eta_i^* \nabla c_i(x_*) + \sum_{j=1}^{p} \mu_j^* \nabla d_j(x_*) = 0, \\ c_i(x_*) \leq 0,\, \eta_i^* \geq 0,\, c_i(x_*)\eta_i^* = 0,\, \forall\, i \in [m] \\ d_j(x_*) = 0,\, \mu_j^* \in \mathbb{R},\, \forall\, j \in [p]. \end{cases} \tag{10}$$

(10) is exactly the KKT conditions of (1) which are satisfied by $(x_*, \{\eta_i^*\}, \{\mu_j^*\})$ and therefore, $x_*$ is a stationary point of (1). $\qquad\square$

Before proving Theorem 4, we need a result to test the closure of $\mathcal{A}_{cccp}$. The following result from [12, Proposition 7] shows that the minimization of a continuous function forms a closed point-to-set map. A similar sufficient condition is also provided in [29, Equation 10].

**Lemma 6** ([12]). *Given a real-valued continuous function $h$ on $X \times Y$, define the point-to-set map $\Psi : X \rightarrow \mathscr{P}(Y)$ by*

$$
\begin{aligned}
\Psi(x) &= \arg \min_{y' \in Y} h(x, y') \\
&= \{y : h(x, y) \leq h(x, y'), \forall \, y' \in Y\}.
\end{aligned} \tag{11}
$$

*Then, $\Psi$ is closed at $x$ if $\Psi(x)$ is nonempty.*

We are now ready to prove Theorem 4.

*Proof of Theorem 4.* The assumption of $\mathcal{A}_{cccp}$ being uniformly compact on $\Omega$ ensures that condition (1) in Theorem 2 is satisfied. Let $\Gamma$ be the set of all generalized fixed points of $\mathcal{A}_{cccp}$ and let $\phi = f = u - v$. Because of the descent property in (5), condition (2) in Theorem 2 is satisfied. By our assumption on $u$ and $v$, we have $g(x, y) = u(x) - v(y) - (x - y)^T \nabla v(y)$ is continuous in $x$ and $y$. Therefore, by Lemma 6, the assumption of non-emptiness of $\mathcal{A}_{cccp}(x)$ for any $x \in \Omega$ ensures that $\mathcal{A}_{cccp}$ is closed on $\Omega$ and so satisfies condition (3) in Theorem 2. Therefore, by Theorem 2, all the limit points of $\{x^{(l)}\}_{l=0}^{\infty}$ are the generalized fixed points of $\mathcal{A}_{cccp}$ and $\lim_{l \rightarrow \infty}(u(x^{(l)}) - v(x^{(l)})) = u(x_*) - v(x_*)$, where $x_*$ is some generalized fixed point of $\mathcal{A}_{cccp}$. By Lemma 5, since the generalized fixed points of $\mathcal{A}_{cccp}$ are stationary points of (1), the result follows. $\quad\square$

**Remark 7.** *If $\Omega$ is compact, then $\mathcal{A}_{cccp}$ is uniformly compact on $\Omega$. In addition, since $u$ is continuous on $\Omega$, by the Weierstrass theorem[4] [21], it is clear that $\mathcal{A}_{cccp}(x)$ is nonempty for any $x \in \Omega$ and therefore is also closed on $\Omega$. This means, when $\Omega$ is compact, the result in Theorem 4 follows trivially from Theorem 2.*

In Theorem 4, we considered the generalized fixed points of $\mathcal{A}_{cccp}$. The disadvantage with this case is that it does not rule out "oscillatory" behavior [20]. To elaborate, we considered $\{x_*\} \subset \mathcal{A}_{cccp}(x_*)$. For example, let $\Omega_0 = \{x_1, x_2\}$ and let $\mathcal{A}_{cccp}(x_1) = \mathcal{A}_{cccp}(x_2) = \Omega_0$ and $u(x_1) - v(x_1) = u(x_2) - v(x_2) = 0$. Then the sequence $\{x_1, x_2, x_1, x_2, \ldots\}$ could be generated by $\mathcal{A}_{cccp}$, with the convergent subsequences converging to the generalized fixed points $x_1$ and $x_2$. Such an oscillatory behavior can be avoided if we allow $\mathcal{A}_{cccp}$ to have fixed points instead of generalized fixed points. With appropriate assumptions on $u$ and $v$, the following stronger result can be obtained on the convergence of CCCP through Theorem 3.

**Theorem 8** (Global convergence of CCCP$-$II). *Let $u$ and $v$ be strictly convex, differentiable functions defined on $\mathbb{R}^n$. Also assume $\nabla v$ be continuous. Let $\{x^{(l)}\}_{l=0}^{\infty}$ be any sequence generated by $\mathcal{A}_{cccp}$ defined by (9). Suppose $\mathcal{A}_{cccp}$ is uniformly compact on $\Omega$ and $\mathcal{A}_{cccp}(x)$ is nonempty for any $x \in \Omega$. Then, assuming suitable constraint qualification, all the limit points of $\{x^{(l)}\}_{l=0}^{\infty}$ are stationary points of the d.c. program in (1), $u(x^{(l)}) - v(x^{(l)}) \rightarrow u(x_*) - v(x_*) =: f^*$ as $l \rightarrow \infty$, for some stationary point $x_*$, $\|x^{(l+1)} - x^{(l)}\| \rightarrow 0$, and either $\{x^{(l)}\}_{l=0}^{\infty}$ converges or the set of limit points of $\{x^{(l)}\}_{l=0}^{\infty}$ is a connected and compact subset of $\mathscr{S}(f^*)$, where $\mathscr{S}(a) := \{x \in \mathscr{S} : u(x) - v(x) = a\}$ and $\mathscr{S}$ is the set of stationary points of (1). If $\mathscr{S}(f^*)$ is finite, then any sequence $\{x^{(l)}\}_{l=0}^{\infty}$ generated by $\mathcal{A}_{cccp}$ converges to some $x_*$ in $\mathscr{S}(f^*)$.*

*Proof.* Since $u$ and $v$ are strictly convex, the strict descent property in (8) holds and therefore $\mathcal{A}_{cccp}$ is strictly monotonic with respect to $f$. Under the assumptions made about $\mathcal{A}_{cccp}$, Theorem 3 can be invoked, which says that all the limit points of $\{x^{(l)}\}_{l=0}^{\infty}$ are fixed points of $\mathcal{A}_{cccp}$, which either converge or form a connected compact set. From Lemma 5, the set of fixed points of $\mathcal{A}_{cccp}$ are already in the set of stationary points of (1) and the desired result follows from Theorem 3. $\quad\square$

Theorems 4 and 8 answer the questions that we raised in Section 1. These results explicitly provide sufficient conditions on $u$, $v$, $\{c_i\}$ and $\{d_j\}$ under which the CCCP algorithm finds a stationary point of (1) along with the convergence of the sequence generated by the algorithm. From Theorem 8, it should be clear that convergence of $f(x^{(l)})$ to $f^*$ does not automatically imply the convergence of $x^{(l)}$ to $x_*$. The convergence in the latter sense requires more stringent conditions like the finiteness of the set of stationary points of (1) that assume the value of $f^*$.

### 4.1 Extensions

So far, we have considered d.c. programs where the constraint set is convex. Let us consider a general d.c. program given by

$$\min_x \quad u_0(x) - v_0(x)$$
$$\text{s.t.} \quad u_i(x) - v_i(x) \leq 0, \, i \in [m], \tag{12}$$

where $\{u_i\}$, $\{v_i\}$ are real-valued convex and differentiable functions defined on $\mathbb{R}^n$. While dealing with kernel methods for missing variables, [26] encountered a problem of the form in (12) for which they proposed a *constrained concave-convex procedure* given by

$$x^{(l+1)} \in \arg\min_x \quad u_0(x) - \widehat{v_0}(x; x^{(l)})$$
$$\text{s.t.} \quad u_i(x) - \widehat{v_i}(x; x^{(l)}) \leq 0, \, i \in [m], \tag{13}$$

where $\widehat{v_i}(x; x^{(l)}) := v_i(x^{(l)}) + (x - x^{(l)})^T \nabla v_i(x^{(l)})$. Note that, similar to CCCP, the algorithm in (13) is a sequence of convex programs. Though [26, Theorem 1] have provided a convergence analysis for the algorithm in (13), it is however not complete due to the fact that the convergence of $\{x^{(l)}\}_{l=0}^\infty$ is assumed. In this subsection, we provide its convergence analysis, following an approach similar to what we did for CCCP by considering a point-to-set map, $\mathcal{B}_{ccp}$ associated with the iterative algorithm in (13), where $x^{(l+1)} \in \mathcal{B}_{ccp}(x^{(l)})$. In Theorem 10, we provide the global convergence result for the constrained concave-convex procedure, which is an equivalent version of Theorem 4 for CCCP. We do not provide the stronger version of the result as in Theorem 8 as it can be obtained by assuming strict convexity of $u_0$ and $v_0$. Before proving Theorem 10, we need an equivalent version of Lemma 5 which we provide below.

**Lemma 9.** *Suppose $x_*$ is a generalized fixed point of $\mathcal{B}_{ccp}$ and assume that constraints in (13) are qualified at $x_*$. Then, $x_*$ is a stationary point of the program in (12).*

*Proof.* Based on the assumptions $x_* \in \mathcal{B}_{ccp}(x_*)$ and the constraint qualification at $x_*$ in (13), there exist Lagrange multipliers $\{\eta_i^*\}_{i=1}^m \subset \mathbb{R}_+$ (for simplicity, we assume all the constraints to be inequality constraints) such that the following KKT conditions hold:

$$\begin{cases} \nabla u_0(x_*) + \sum_{i=1}^m \eta_i^*(\nabla u_i(x_*) - \nabla v_i(x_*)) = \nabla v_0(x_*), \\ u_i(x_*) - v_i(x_*) \leq 0, \, \eta_i^* \geq 0, \, i \in [m], \\ (u_i(x_*) - v_i(x_*))\eta_i^* = 0, \, i \in [m]. \end{cases} \tag{14}$$

which is exactly the KKT conditions for (12) satisfied by $(x_*, \{\eta_i^*\})$ and therefore, $x_*$ is a stationary point of (12). □

**Theorem 10** (Global convergence of constrained CCP). *Let $\{u_i\}$, $\{v_i\}$ be real-valued differentiable convex functions on $\mathbb{R}^n$. Assume $\nabla v_0$ to be continuous. Let $\{x^{(l)}\}_{l=0}^\infty$ be any sequence generated by $\mathcal{B}_{ccp}$ defined in (13). Suppose $\mathcal{B}_{ccp}$ is uniformly compact on $\Omega := \{x : u_i(x) - v_i(x) \leq 0, \, i \in [m]\}$ and $\mathcal{B}_{ccp}(x)$ is nonempty for any $x \in \Omega$. Then, assuming suitable constraint qualification, all the limit points of $\{x^{(l)}\}_{l=0}^\infty$ are stationary points of the d.c. program in (12). In addition $\lim_{l \to \infty}(u_0(x^{(l)}) - v_0(x^{(l)})) = u_0(x_*) - v_0(x_*)$, where $x_*$ is some stationary point of $\mathcal{B}_{ccp}$.*

*Proof.* The proof is very similar to that of Theorem 4 wherein we check whether $\mathcal{B}_{ccp}$ satisfies the conditions of Theorem 2 and then invoke Lemma 9. The assumptions mentioned in the statement of the theorem ensure that conditions (1) and (3) in Theorem 2 are satisfied. [26, Theorem 1] has proved the descent property, similar to that of (5), which simply follows from the linear majorization idea and therefore the descent property in condition (2) of Theorem 2 holds. Therefore, the result follows from Theorem 2 and Lemma 9. □

## 5 On the local convergence of CCCP: An open problem

The study so far has been devoted to the global convergence analysis of CCCP and the constrained concave-convex procedure. As mentioned before, we say an algorithm is globally convergent if for *any* chosen starting point, $x_0$, the sequence $\{x_k\}_{k=0}^\infty$ generated by $x_{k+1} \in \mathcal{A}(x_k)$ converges to a point for which a necessary condition of optimality holds. In the results so far, we have shown

that all the limit points of any sequence generated by CCCP (*resp.* its constrained version) are the stationary points (local extrema or saddle points) of the program in (1) (*resp.* (12)). Suppose, if $x_0$ is chosen such that it lies in an $\epsilon$-neighborhood around a local minima, $x_*$, then will the CCCP sequence converge to $x_*$? If so, what is the rate of convergence? This is the question of *local convergence* that needs to be addressed.

[24] has studied the local convergence of bound optimization algorithms (of which CCCP is an example) to compare the rate of convergence of such methods to that of gradient and second-order methods. In their work, they considered the unconstrained version of CCCP with $\mathcal{A}_{cccp}$ to be a point-to-point map that is differentiable. They showed that depending on the curvature of $u$ and $v$, CCCP will exhibit either quasi-Newton behavior with fast, typically superlinear convergence or extremely slow, first-order convergence behavior. However, extending these results to the constrained setup as in (2) is not obvious. The following result due to Ostrowski which can be found in [23, Theorem 10.1.3] provides a way to study the local convergence of iterative algorithms.

**Proposition 11** (Ostrowski). *Suppose that $\Psi : U \subset \mathbb{R}^n \to \mathbb{R}^n$ has a fixed point $x_* \in int(U)$ and $\Psi$ is Fréchet-differentiable at $x_*$. If the spectral radius of $\Psi'(x_*)$ satisfies $\rho(\Psi'(x_*)) < 1$, and if $x_0$ is sufficiently close to $x_*$, then the iterates $\{x_k\}$ defined by $x_{k+1} = \Psi(x_k)$ all lie in $U$ and converge to $x_*$.*

Few remarks are in place regarding the usage of Proposition 11 to study the local convergence of CCCP. Note that Proposition 11 treats $\Psi$ as a point-to-point map which can be obtained by choosing $u$ and $v$ to be strictly convex so that $x^{(l+1)}$ is the unique minimizer of (2). $x_*$ in Proposition 11 can be chosen to be a local minimum. Therefore, the desired result of local convergence with at least linear rate of convergence is obtained if we show that $\rho(\Psi'(x_*)) < 1$. However, currently we are not aware of a way to compute the differential of $\Psi$ and, moreover, to impose conditions on the functions in (2) so that $\Psi$ is a differentiable map. This is an open question coming out of this work.

On the other hand, the local convergence behavior of DCA has been proved for two important classes of d.c. programs: (i) the trust region subproblem [9] (minimization of a quadratic function over a Euclidean ball) and (ii) nonconvex quadratic programs [8]. We are not aware of local optimality results for general d.c. programs using DCA.

# 6 Conclusion & Discussion

The concave-convex procedure (CCCP) is widely used in machine learning. In this work, we analyze its global convergence behavior by using results from the global convergence theory of iterative algorithms. We explicitly mention the conditions under which any sequence generated by CCCP converges to a stationary point of a d.c. program with convex constraints. The proposed approach allows an elegant and direct proof and is fundamentally different from the highly technical proof for the convergence of DCA, which implies convergence for CCCP. It illustrates the power and generality of Zangwill's global convergence theory as a framework for proving the convergence of iterative algorithms. We also briefly discuss the local convergence of CCCP and present an open question, the settlement of which would address the local convergence behavior of CCCP.

**Acknowledgments**

The authors thank anonymous reviewers for their constructive comments. They wish to acknowledge support from the National Science Foundation (grant DMS-MSPA 0625409), the Fair Isaac Corporation and the University of California MICRO program.

## Footnotes

[1]$x_*$ is said to be a stationary point of a constrained optimization problem if it satisfies the corresponding Karush-Kuhn-Tucker (KKT) conditions. Assuming constraint qualification, KKT conditions are necessary for the local optimality of $x_*$. See [2, Section 11.3] for details.

[2]Note that depending on the objective and constraints, the minimizer of the CCCP algorithm in (2) need not be unique. Therefore, the algorithm takes $x^{(l)}$ as its input and returns a set of minimizers from which an element, $x^{(l+1)}$ is chosen. Hence the notion of point-to-set maps appear naturally in such iterative algorithms.

[3]Assuming that for every $x \in \Omega$, the set $H(x) := \{y \,:\, u(y) - u(x) \leq v(y) - v(x),\, y \in \mathcal{A}_{cccp}(\Omega)\}$ is bounded is also sufficient for the result to hold.

[4]Weierstrass theorem states: If $f$ is a real continuous function on a compact set $K \subset \mathbb{R}^n$, then the problem $\min\{f(x) : x \in K\}$ has an optimal solution $x^* \in K$.

# References

[1] D. Böhning and B. G. Lindsay. Monotonicity of quadratic-approximation algorithms. *Annals of the Institute of Statistical Mathematics*, 40(4):641–663, 1988.

[2] J. F. Bonnans, J. C. Gilbert, C. Lemaréchal, and C. A. Sagastizábal. *Numerical Optimization: Theoretical and Practical Aspects*. Springer-Verlag, 2006.

[3] P. S. Bradley and O. L. Mangasarian. Feature selection via concave minimization and support vector machines. In *Proc. 15th International Conf. on Machine Learning*, pages 82–90. Morgan Kaufmann, San Francisco, CA, 1998.

[4] E. J. Candes, M. Wakin, and S. Boyd. Enhancing sparsity by reweighted $\ell_1$ minimization. *J. Fourier Anal. Appl.*, 2007. To appear.

[5] R. Collobert, F. Sinz, J. Weston, and L. Bottou. Large scale transductive SVMs. *Journal of Machine Learning Research*, 7:1687–1712, 2006.

[6] J. deLeeuw. Applications of convex analysis to multidimensional scaling. In J. R. Barra, F. Brodeau, G. Romier, and B. Van Cutsem, editors, *Recent advantages in Statistics*, pages 133–146, Amsterdam, The Netherlands, 1977. North Holland Publishing Company.

[7] A. P. Dempster, N. M. Laird, and D. B. Rubin. Maximum likelihood from incomplete data via the EM algorithm. *J. Roy. Stat. Soc. B*, 39:1–38, 1977.

[8] T. Pham Dinh and L. T. Hoai An. Convex analysis approach to d.c. programming: Theory, algorithms and applications. *Acta Mathematica Vietnamica*, 22(1):289–355, 1997.

[9] T. Pham Dinh and L. T. Hoai An. D.c. optimization algorithms for solving the trust region subproblem. *SIAM Journal of Optimization*, 8:476–505, 1998.

[10] C. B. Do, Q. V. Le, C. H. Teo, O. Chapelle, and A. J. Smola. Tighter bounds for structured estimation. In *Advances in Neural Information Processing Systems 21*, 2009. To appear.

[11] G. Fung and O. L. Mangasarian. Semi-supervised support vector machines for unlabeled data classification. *Optimization Methods and Software*, 15:29–44, 2001.

[12] A. Gunawardana and W. Byrne. Convergence theorems for generalized alternating minimization procedures. *Journal of Machine Learning Research*, 6:2049–2073, 2005.

[13] W. J. Heiser. Correspondence analysis with least absolute residuals. *Comput. Stat. Data Analysis*, 5:337–356, 1987.

[14] P. J. Huber. *Robust Statistics*. John Wiley, New York, 1981.

[15] D. R. Hunter and K. Lange. A tutorial on MM algorithms. *The American Statistician*, 58:30–37, 2004.

[16] D. R. Hunter and R. Li. Variable selection using MM algorithms. *Annals of Statistics*, 33:1617–1642, 2005.

[17] K. Lange, D. R. Hunter, and I. Yang. Optimization transfer using surrogate objective functions with discussion. *Journal of Computational and Graphical Statistics*, 9(1):1–59, 2000.

[18] D. D. Lee and H. S. Seung. Algorithms for non-negative matrix factorization. In T.K. Leen, T.G. Dietterich, and V. Tresp, editors, *Advances in Neural Information Processing Systems 13*, pages 556–562. MIT Press, Cambridge, 2001.

[19] X.-L. Meng. Discussion on "optimization transfer using surrogate objective functions". *Journal of Computational and Graphical Statistics*, 9(1):35–43, 2000.

[20] R. R. Meyer. Sufficient conditions for the convergence of monotonic mathematical programming algorithms. *Journal of Computer and System Sciences*, 12:108–121, 1976.

[21] M. Minoux. *Mathematical Programming: Theory and Algorithms*. John Wiley & Sons Ltd., 1986.

[22] J. Neumann, C. Schnörr, and G. Steidl. Combined SVM-based feature selection and classification. *Machine Learning*, 61:129–150, 2005.

[23] J. M. Ortega and W. C. Rheinboldt. *Iterative Solution of Nonlinear Equations in Several Variables*. Academic Press, New York, 1970.

[24] R. Salakhutdinov, S. Roweis, and Z. Ghahramani. On the convergence of bound optimization algorithms. In *Proc. 19th Conference in Uncertainty in Artificial Intelligence*, pages 509–516, 2003.

[25] F. Sha, Y. Lin, L. K. Saul, and D. D. Lee. Multiplicative updates for nonnegative quadratic programming. *Neural Computation*, 19:2004–2031, 2007.

[26] A. J. Smola, S. V. N. Vishwanathan, and T. Hofmann. Kernel methods for missing variables. In *Proc. of the Tenth International Workshop on Artificial Intelligence and Statistics*, 2005.

[27] B. K. Sriperumbudur, D. A. Torres, and G. R. G. Lanckriet. Sparse eigen methods by d.c. programming. In *Proc. of the $24^{th}$ Annual International Conference on Machine Learning*, 2007.

[28] L. Wang, X. Shen, and W. Pan. On transductive support vector machines. In J. Verducci, X. Shen, and J. Lafferty, editors, *Prediction and Discovery*. American Mathematical Society, 2007.

[29] C. F. J. Wu. On the convergence properties of the EM algorithm. *Annals of Statistics*, 11(1):95–103, 1983.

[30] A. L. Yuille and A. Rangarajan. The concave-convex procedure. *Neural Computation*, 15:915–936, 2003.

[31] W. I. Zangwill. *Nonlinear Programming: A Unified Approach*. Prentice-Hall, Englewood Cliffs, N.J., 1969.

